# A Real Time Clustering CMOS Neural Engine

**T. Serrano-Gotarredona, B. Linares-Barranco, and J. L. Huertas**

Dept. of Analog Design, National Microelectronics Center (CNM), Ed. CICA, Av. Reina Mercedes s/n, 41012 Sevilla, SPAIN. Phone: (34)-5-4239923, Fax: (34)-5-4624506, E-mail: bernabe@cnm.us.es

## Abstract

We describe an analog VLSI implementation of the ART1 algorithm (Carpenter, 1987). A prototype chip has been fabricated in a standard low cost 1.5μm double-metal single-poly CMOS process. It has a die area of $1cm^2$ and is mounted in a 120-pins PGA package. The chip realizes a modified version of the original ART1 architecture. Such modification has been shown to preserve all computational properties of the original algorithm (Serrano, 1994a), while being more appropriate for VLSI realizations. The chip implements an ART1 network with 100 *F1* nodes and 18 *F2* nodes. It can therefore cluster 100 binary pixels input patterns into up to 18 different categories. Modular expansibility of the system is possible by assembling an *N×M* array of chips without any extra interfacing circuitry, resulting in an *F1* layer with 100×*N* nodes, and an *F2* layer with 18×*M* nodes. Pattern classification is performed in less than 1.8μs, which means an equivalent computing power of $2.2×10^9$ connections and connection-updates per second. Although internally the chip is analog in nature, it interfaces to the outside world through digital signals, thus having a true asynchrounous digital behavior. Experimental chip test results are available, which have been obtained through test equipments for digital chips.

## 1    INTRODUCTION

The original ART1 algorithm (Carpenter, 1987) proposed in 1987 is a massively parallel architecture for a self-organizing neural binary-pattern recognition machine. In response to arbitrary orderings of arbitrarily many and complex binary input patterns, ART1 is

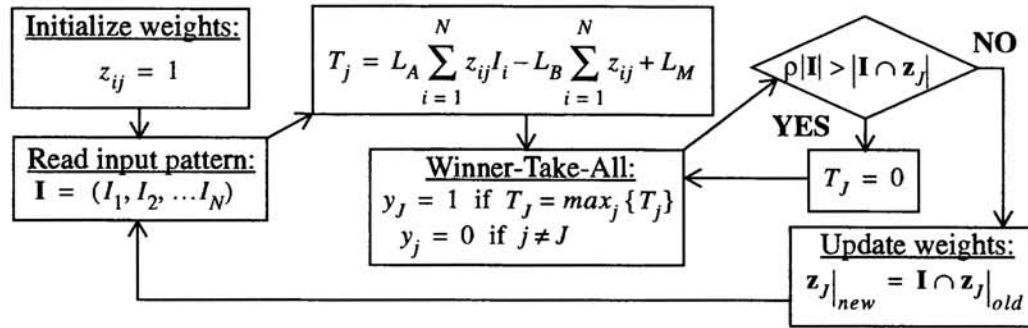

**Fig. 1: Modified *Fast Learning* or *Type-3* ART1
implementation algorithm**

capable of learning, in an unsupervised way, stable recognition codes. The ART1 architecture is described by a set of Short Term Memory (STM) and another set of Long Term Memory (LTM) time domain nonlinear differential equations. It is valid to assume that the STM equations settle much faster (instantaneously) than the LTM equations, so that the STM differential equations can be substituted by nonlinear algebraic equations that describe the steady-state of the STM differential equations. Furthermore, in the fast-learning mode (Carpenter, 1987), the LTM differential equations are as well substituted by their corresponding steady-state nonlinear algebraic equations. This way, the ART1 architecture can be behaviorally modelled by the sequential application of nonlinear algebraic equations. Three different levels of ART1 implementations (both in software and in hardware) can therefore be distinguished:

*Type-1: Full Model Implementation:* both STM and LTM time-domain differential equations are realized. This implementation is the most expensive (both in software and in hardware), and requires a large amount of computational power.

*Type-2: STM steady-state Implementation:* only the LTM time-domain differential equations are implemented. The STM behavior is governed by nonlinear algebraic equations. This implementation requires less resources than the previous one. However, a proper sequencing of STM events has to be introduced artificially, which is architecturally implicit in the *Type-1* implementation.

*Type-3: Fast Learning Implementation:* STM and LTM is implemented with algebraic equations. This implementation is computationally the less expensive one. In this case an artificial sequencing of STM and LTM events has to be done.

The implementation presented in this paper realizes a modified version of the original ART1 *Type-3* algorithm, more suitable for VLSI implementations. Such modified ART1 system has been shown to preserve all computational properties of the original ART1 architecture (Serrano, 1994a). The flow diagram that describes the modified ART1 architecture is shown in Fig. 1. Note that there is only one binary-valued weight template ($z_{ij}$), instead of the two weight templates (one binary-valued and the other real-valued) of the original ART1. For a more detailed discussion of the modified ART1 algorithm refer to (Serrano, 1994a, 1994b).

In the next Section we will provide an analog current-mode based circuit that implements in hardware the flow diagram of Fig. 1. Note that, although internally this circuit is analog in nature, from its input and output signals point of view it is a true asynchronous digital

circuit, easy to interface with any conventional digital machine. Finally, in Section 3 we will provide experimental results measured from the chip using a digital data acquisition test equipment.

## 2   CIRCUIT DESCRIPTION

The ART1 chip reported in this paper has an *F1* layer with 100 neurons and an *F2* layer with 18 neurons. This means that it can handle binary input patterns of 100 pixels each, and cluster them into up to 18 different categories, according to a digitally adjustable vigilance parameter $\rho$. The circuit architecture of the chip is shown in Fig. 2(a). It consists of an array of 18×100 synapses, a 1×100 array of "vigilance synapses", a unity gain 18-outputs current mirror, an adjustable gain 18-outputs current mirror (with $\rho$=0.0, 0.1, ... 0.9)[1], 18 current-comparator-controlled switches and an 18-input-currents Winner-Take-All (WTA) (Serrano, 1994b). The inputs to the circuit are the 100 binary digital input voltages $I_i$, and the outputs of the circuit are the 18 digital output voltages $y_j$. External control signals allow to change parameters $\rho$, $L_A$, $L_B$, and $L_M$. Also, extra circuitry has been added for reading the internal weights $z_{ij}$ while the system is learning.

Each row of synapses generates two currents,

$$T_j = L_A \sum_{i=1}^{100} z_{ij} I_i - L_B \sum_{i=1}^{100} z_{ij} + L_M$$

$$V_j = L_A \sum_{i=1}^{100} z_{ij} I_i \tag{1}$$

while the row of the "vigilance synapses" generates the current

$$V_\rho = L_A \sum_{i=1}^{100} I_i \tag{2}$$

Each of the current comparators compares the current $V_j$ versus $\rho V_\rho$, and allows current $T_j$ to reach the WTA only if $\rho V_\rho \le V_j$. This way competition and vigilance occur simultaneously and in parallel, speeding up significantly the search process.

Fig. 2(b) shows the content of a synapse in the 18x100 array. It consists of three current sources with switches, two digital AND gates and a flip-flop. Each synapse receives two input voltages $I_i$ and $y_j$, and two global control voltages $\phi_l$ (to enable/disable learning) and *reset* (to initialize all weights $z_{ij}$ to '1'). Each synapse generates two currents $L_A I_i z_{ij} - L_B z_{ij}$ and $L_A I_i z_{ij}$, which will be summed up for all the synapses in the same row to generate the currents $T_j$ and $V_j$. If learning is enabled ($\phi_l = 1$) the value of $z_{ij}$ will change to $I_i z_{ij}$ if $y_j = 1$. The "vigilance synapses" consist each of a current-source of value $L_A$ with a switch controlled by the input voltage $I_i$. The current comparators are those proposed in (Domínguez-Castro, 1992), the WTA used is reported in (Lazzaro, 1989), and the digitally adjustable current mirror is based on (Loh, 1989), while its continuous gain fine tuning mechanism has been taken from (Adams, 1991).

---

1. An additional pin of the chip can fine-tune $\rho$ between 0.9 and 1.0.

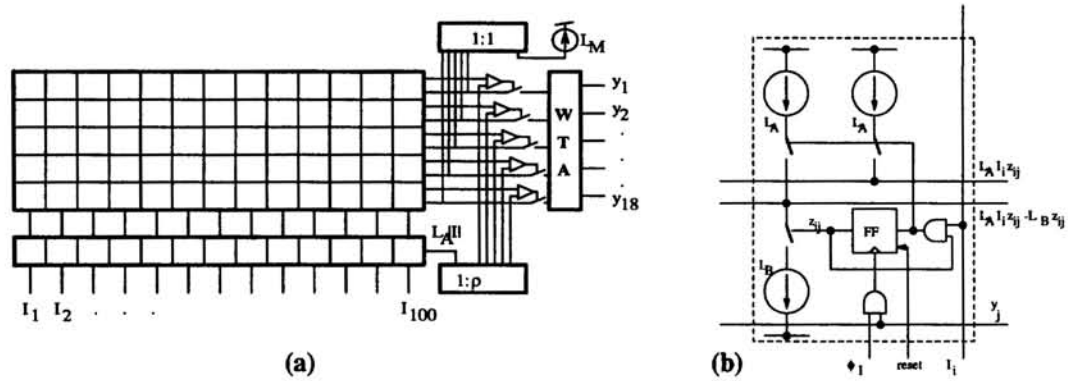

Fig. 2: (a) System Diagram of Current-Mode ART1 Chip, (b) Circuit Diagram of Synapse

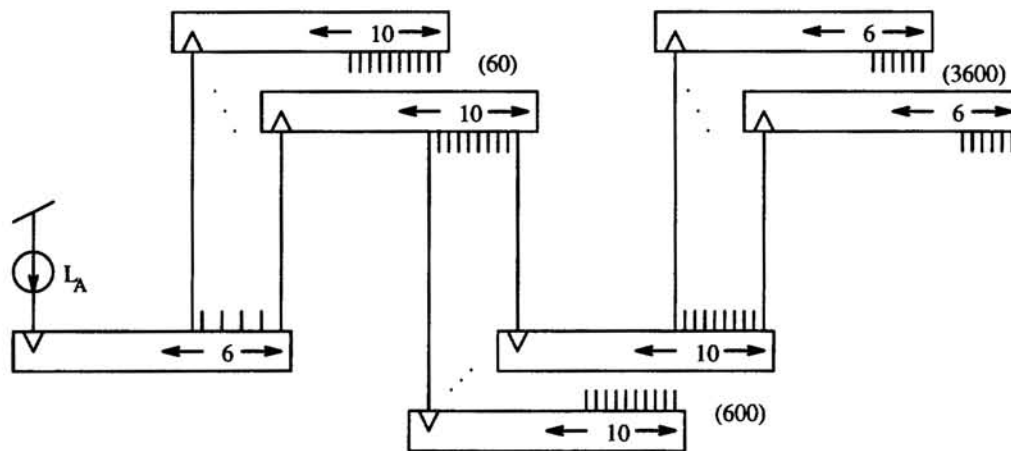

Fig. 3: Tree based current-mirror scheme for matched current sources

The circuit has been designed in such a way that the WTA operates with a precision around 1.5% (~6 bits). This means that all $L_A$ and $L_B$ current sources have to match within an error of less than that. From a circuit implementation point of view this is not easy to achieve, since there are 5500 current sources spread over a die area of $1 cm^2$. Typical mismatch between reasonable size MOS transistors inside such an area extension can be expected to be above 10% (Pelgrom, 1989). To overcome this problem we implemented a tree-based current mirror scheme as is shown in Fig. 3. Starting from a unique current reference, and using high-precision 10(or less)-outputs current mirrors (each yielding a precision around 0.2%), only up to four cascades are needed. This way, the current mismatch attained at the synapse current sources was around 1% for currents between $L_{A/B} = 5\mu A$ and $L_{A/B} = 10\mu A$. This is shown in Fig. 4, where the measured dc output current-error (in %) versus input current of the tree based configuration for 18 of the 3600 $L_A$ synapse sources is depicted.

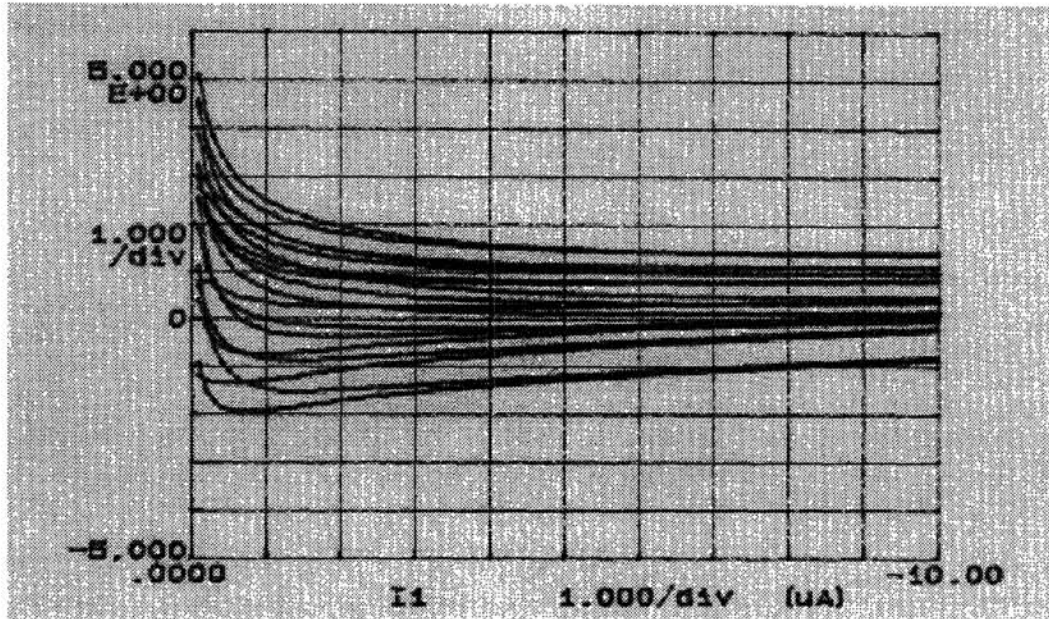

**Fig. 4: Measured current mirror cascade missmatch (1%/div) for $L_A$ for currents below 10μA**

## 3 EXPERIMENTAL RESULTS

Fig. 5 shows a microphotograph of a prototype chip fabricated in a standard digital double-metal, single-poly 1.5μm low cost CMOS process. The chip die area is $1cm^2$, and it is mounted in a 120-pins PGA package. Fig. 6 shows a typical training sequence accomplished by the chip and obtained experimentally using a test equipment for digital chips. The only task performed by the test equipment was to provide the input data patterns **I** (first column in Fig. 6), detect which of the output nodes became '1' (pattern with a vertical bar to its right), and extract the learned weights. Each 10×10 square in Fig. 6 represents either a 100-pixels input vector **I**, or one row of 100-pixels synaptic weights $\mathbf{z}_j \equiv (z_{1j}, z_{2j}, ...z_{100j})$ . Each row of squares in Fig. 6 represents the input pattern (first square) and the 18 vectors $\mathbf{z}_j$ after learning has been performed for this input pattern. The sequence shown in Fig. 6 has been obtained for $\rho = 0.7$, $L_A = 10\mu A$, $L_B = 9.5\mu A$, and $L_M = 950\mu A$. Only two iterations of input patterns presentations were necessary, in this case, for the system to learn and self-organize in response to these 18 input patterns.

The last row in Fig. 6 shows the final learned templates. Fig. 7 shows final learned templates for different values of $\rho$. The numbers below each square indicate the input patterns that have been clustered into each $\mathbf{z}_j$ category.

Delay time measurements have been performed for the feedforward action of the chip (establishment of currents $T_j$, $V_j$, and $V_\rho$, and their competitions until the WTA settles), and for the updating of weights. The feedforward delay is pattern and bias currents ($L_A$, $L_B$, $L_M$) dependent, but has been measured to be always below 1.6μs. The learning time is constant and is around 180ns. Therefore, throughput time is less than 1.8μs. A digital neuroprocessor able to perform $a$ connections/s, $b$ connection-updates/s, and with a dedicated WTA section with a $c$ seconds delay, must satisfy

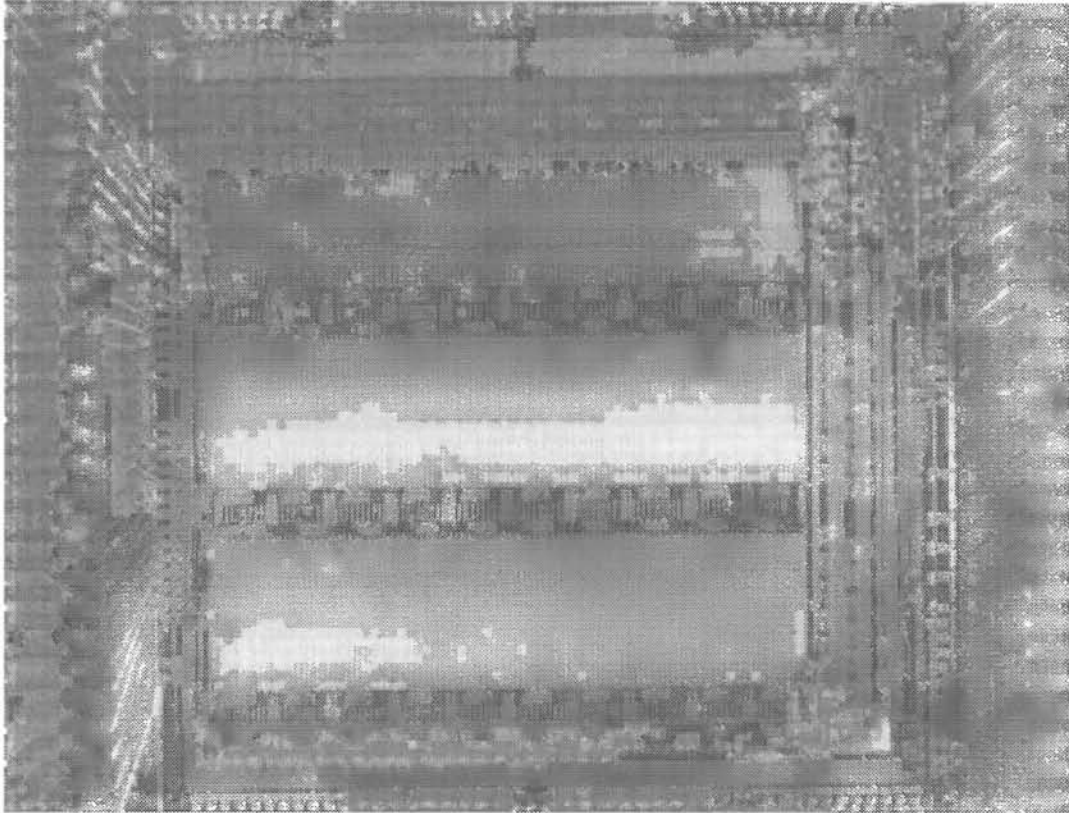

**Fig. 5: Microphotograph of ART1 chip**

$$\frac{3700}{a} + \frac{100}{b} + c = 1.8\mu s \tag{3}$$

to meet the performance of our prototype chip. If $a = b$ and $c = 100ns$, the equivalent speed would be $a = b = 2.2 \times 10^9$ connections and connection-updates per second.

## 4 CONCLUSIONS

A high speed analog current-mode categorizer chip has been built using a standard low cost digital CMOS process. The high performance of the chip is achieved thanks to a simplification of the original ART1 algorithm. The simplifications introduced are such that all the original computational properties are preserved. Experimental chip test results are provided.

Fig. 6: Test sequence obtained experimentally for $\rho=0.7$, $L_A=10\mu A$, $L_B=9.5\mu A$, and $L_M=950\mu A$

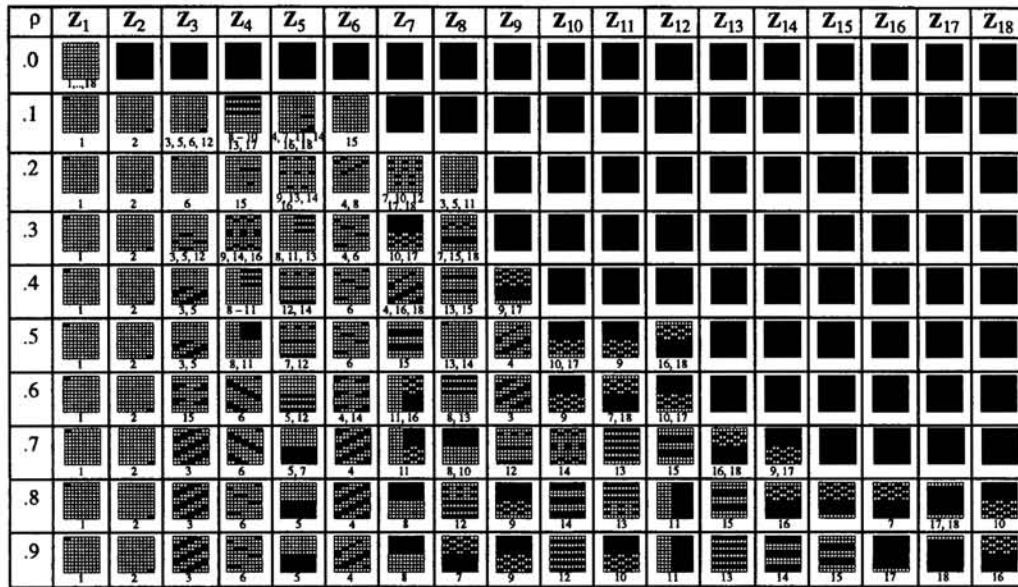

**Fig. 7: Categorization of the input patterns for $L_A=3.2\mu A$, $L_B=3.0\mu A$, $L_M=400\mu A$, and different values of $\rho$**

# References

W. J. Adams and J. Ramímez-Angulo. (1991) "Extended Transconductance Adjustment/Linearisation Technique," *Electronics Letters*, vol. 27, No. 10, pp. 842-844, May 1991.

G. A. Carpenter and S. Grossberg. (1987) "A Massively Parallel Architecture for a Self-Organizing Neural Pattern Recognition Machine," *Computer Vision, Graphics, and Image Processing*, vol. 37, pp. 54-115, 1987.

R. Domínguez-Castro, A. Rodríguez-Vázquez, F. Medeiro, and J. L. Huertas. (1992) "High Resolution CMOS Current Comparators," *Proc. of the 1992 European Solid-State Circuits Conference (ESSCIRC'92)*, pp. 242-245, 1992.

J. Lazzaro, R. Ryckebusch, M. A. Mahowald, and C. Mead. (1989) "Winner-Take-All Networks of O(n) Complexity," in *Advances in Neural Information Processing Systems*, vol. 1, D. S. Touretzky (Ed.), Los Altos, CA: Morgan Kaufmann, 1989, pp. 703-711.

K. Loh, D. L. Hiser, W. J. Adams, and R. L. Geiger. (1989) "A Robust Digitally Programmable and Reconfigurable Monolithic Filter Structure," *Proc. of the 1989 Int. Symp. on Circuits and Systems (ISCAS'89)*, Portland, Oregon, vol. 1, pp. 110-113, 1989.

M. J. Pelgrom, A. C. J. Duinmaijer, and A. P. G. Welbers. (1989) "Matching Properties of MOS Transistors," *IEEE Journal of Solid-State Circuits*, vol. 24, No. 5, pp. 1433-1440, October 1989.

T. Serrano-Gotarredona and B. Linares-Barranco. (1994a) "A Modified ART1 Algorithm more suitable for VLSI Implementations," submitted for publication (journal paper).

T. Serrano-Gotarredona and B. Linares-Barranco. (1994b) "A Real-Time Clustering Microchip Neural Engine," submitted for publication (journal paper).
